# MLP can provably generalise much better than VC-bounds indicate.

**A. Kowalczyk and H. Ferrá**
Telstra Research Laboratories
770 Blackburn Road, Clayton, Vic. 3168, Australia
({a.kowalczyk, h.ferra}@trl.oz.au)

## Abstract

Results of a study of the worst case learning curves for a particular class of probability distribution on input space to MLP with hard threshold hidden units are presented. It is shown in particular, that in the thermodynamic limit for scaling by the number of connections to the first hidden layer, although the true learning curve behaves as $\approx \alpha^{-1}$ for $\alpha \approx 1$, its VC-dimension based bound is trivial ($\equiv 1$) and its VC-entropy bound is trivial for $\alpha \leq 6.2$. It is also shown that bounds following the true learning curve can be derived from a formalism based on the density of error patterns.

## 1 Introduction

The VC-formalism and its extensions link the generalisation capabilities of a binary valued neural network with its *counting function*[1], e.g. via upper bounds implied by VC-dimension or VC-entropy on this function [17, 18]. For linear perceptrons the counting function is constant for almost every selection of a fixed number of input samples [2], and essentially equal to its upper bound determined by VC-dimension and Sauer's Lemma. However, in the case for *multilayer perceptrons* (MLP) the counting function depends essentially on the selected input samples. For instance, it has been shown recently that for MLP with sigmoidal units although the largest number of input samples which can be shattered, i.e. VC-dimension, equals $\Omega(w^2)$ [6], there is always a non-zero probability of finding a $(2w+2)$-element input sample which cannot be shattered, where $w$ is the number of weights in the network [16]. In the case of MLP using Heaviside rather than sigmoidal activations (McCulloch-Pitts neurons), a similar claim can be made: VC-dimension is $\Omega(w_1 log_2 \mathcal{H}_1)$ [13, 15],

where $w_1$ is the number of weights to the first hidden layer of $\mathcal{H}_1$ units, but there is a non-zero probability of finding a sample of size $w_1 + 2$ which cannot be shattered [7, 8]. The results on these "hard to shatter samples" for the two MLP types differ significantly in terms of techniques used for derivation. For the sigmoidal case the result is "existential" (based on recent advances in "model theory") while in the Heaviside case the proofs are constructive, defining a class of probability distributions from which "hard to shatter" samples can be drawn randomly; the results in this case are also more explicit in that a form for the counting function may be given [7, 8].

Can the existence of such hard to shatter samples be essential for generalisation capabilities of MLP? Can they be an essential factor for improvement of theoretical models of generalisation? In this paper we show that at least for the McCulloch-Pitts case with specific (continuous) probability distributions on the input space the answer is "yes". We estimate "directly" the real learning curve in this case and show that its bounds based on VC-dimension or VC-entropy are loose at low learning sample regimes (for training samples having less than $12 \times w_1$ examples) even for the linear perceptron. We also show that a modification to the VC-formalism given in [9, 10] provides a significantly better bound. This latter part is a more rigorous and formal extension and re-interpretation of some results in [11, 12]. All the results are presented in the thermodynamic limit, i.e. for MLP with $w_1 \to \infty$ and training sample size increasing proportionally, which simplifies their mathematical form.

## 2   Overview of the formalism

On a sample space $X$ we consider a class $H$ of binary functions $h : X \to \{0,1\}$ which we shall call *a hypothesis space*. Further we assume that there are given a probability distribution $\mu$ on $X$ and a *target concept* $t : X \to \{0,1\}$. The quadruple $\mathcal{L} = (X, \mu, H, t)$ will be called *a learning system*.

In the usual way, with each hypothesis $h \in H$ we associate *the generalization error* $\epsilon_h \overset{def}{=} \mathbf{E}_X\big[|t(x) - h(x)|\big]$ and *the training error* $\epsilon_{h,\vec{x}} \overset{def}{=} \frac{1}{m} \sum_{i=1}^{m} |t(x_i) - h(x_i)|$ for any *training m-sample* $\vec{x} = (x_1, ..., x_m) \in X^m$.

Given a *learning threshold* $0 \leq \lambda \leq 1$, let us introduce an auxiliary random variable $\epsilon_\lambda^{\max}(\vec{x}) \overset{def}{=} \max\{\epsilon_h \; ; \; h \in H \ \& \ \epsilon_{h,\vec{x}} \leq \lambda\}$ for $\vec{x} \in X^m$, giving the worst generalization error of all hypotheses with training error $\leq \lambda$ on the $m$-sample $\vec{x} \in X^m$. [2] The basic objects of interest in this paper are *the learning curve*[3] defined as

$$\epsilon_\lambda^{wc}(m) \overset{def}{=} \mathbf{E}_{X^m}[\epsilon_\lambda^{\max}(\vec{x})].$$

### 2.1   Thermodynamic limit

Now we introduce the thermodynamic limit of the learning curve. The underlying idea of such asymptotic analysis is to capture the essential features of learning

systems of very large size. Mathematically it turns out that in the thermodynamic limit the functional forms of learning curves simplify significantly and analytic characterizations of these are possible.

We are given a sequence of learning systems, or shortly, $\mathcal{L}_N = (X_N, \mu_N, H_N, t_N)$, $N = 1, 2, ...$ and *a scaling* $N \mapsto \tau_N \in \mathbf{R}^+$, with the property $\tau_N \to \infty$; the scaling can be thought of as a measure of the size (complexity) of a learning system, e.g. VC-dimension of $H_N$. *The thermodynamic limit* of *scaled learning curves* is defined for $\alpha > 0$ as follows [4]

$$\epsilon_{\lambda\infty}^{wc}(\alpha) \stackrel{def}{=} \limsup_{N\to\infty} \epsilon_{\lambda,N}^{wc}(\lfloor \alpha\tau_N \rfloor), \tag{1}$$

Here, and below, the additional subscript $N$ refers to the $N$-th learning system.

## 2.2   Error pattern density formalism

This subsection briefly presents a thermodynamic version of a modified VC formalism discussed previously in [9]; more details and proofs can be found in [10]. The main innovation of this approach comes from splitting error patterns into error shells and using estimates on the size of these error shells rather than the total number of error patterns. We shall see on examples discussed in the following section that this improves results significantly.

The space $\{0, 1\}^m$ of all binary $m$-vectors naturally splits into $m + 1$ *error pattern shells* $\mathcal{E}_i^m$, $i = 0, 1, ..., m$, with the $i$-th shell composed of all vectors with exactly $i$ entries equal to 1 . For each $h \in H$ and $\vec{x} = (x_1, ..., x_m) \in X^m$, let $\vec{v}_h(\vec{x}) \in \{0, 1\}^m$ denote a vector (*error pattern*) having 1 in the $j$-th position if and only if $h(x_j) \neq t(x_j)$. As the $i$-th error shell has $\binom{m}{i}$ elements, *the average error pattern density* falling into this error shell is

$$\Delta_i^m \stackrel{def}{=} \binom{m}{i}^{-1} \mathbf{E}_{X^m}[\#(\{\vec{v}_h(\vec{x}) \; ; \; h \in H\} \cap \mathcal{E}_i^m)] \quad (i = 0, 1, ..., m), \tag{2}$$

where $\#$ denotes the cardinality of a set [5] .

**Theorem 1** *Given a sequence of learning systems* $\mathcal{L}_N = (X_N, \mu_N, H_N, t_N)$, *a scaling* $\tau_N$ *and a function* $\varphi : \mathbf{R}^+ \times (0, 1) \to \mathbf{R}^+$ *such that*

$$\ln\left(\Delta_{i,N}^m\right) \leq -\tau_N \varphi\left(\frac{m}{\tau_N}, \frac{i}{m}\right) + o(\tau_N), \tag{3}$$

*for all* $m, N = 1, 2, ..., 0 \leq i \leq m$.

*Then*

$$\epsilon_{\lambda\infty}^{wc}(\alpha) \leq \epsilon_{\lambda\beta}(\alpha), \tag{4}$$

for any $0 \leq \lambda \leq 1$ and $\alpha, \beta > 0$, where

$$\epsilon_{\lambda\beta}(\alpha) \overset{def}{=} \max \left\{ \epsilon \in (0,1) \; ; \; \exists_{\substack{0 \leq y \leq \lambda \\ \epsilon \leq x \leq 1}} \alpha(\mathcal{H}(y) + \beta \mathcal{H}(x)) - \varphi\left(\alpha + \alpha\beta, \frac{y + \beta x}{1 + \beta}\right) \geq 0 \right\}$$

and $\mathcal{H}(y) \overset{def}{=} -y \ln y - (1-y) \ln(1-y)$ denotes the entropy function.

## 3  Main results : applications of the formalism

### 3.1  VC-bounds

We consider a learning sequence $\mathcal{L}_N = (X_N, \mu_N, H_N, t_N)$, $t_N \in H_N$ (realisable case) and the scaling of this sequence by VC-dimension [17], i.e. we assume $\tau_N = d_{VC}(H_N) \to \infty$. The following bounds for the $N$-th learning system can be derived for $\lambda = 0$ (consistent learning case) [1, 17]:

$$\epsilon_{0,N}^{wc}(m) \leq \int_0^1 \min\left(1, 2^{2-m\epsilon/2} \left(\frac{2em}{d_{VC}(H_N)}\right)^{d_{VC}(H_N)}\right) d\epsilon. \tag{5}$$

In the thermodynamic limit, i.e. as $N \to \infty$, we get for any $\alpha > 1/e$

$$\epsilon_{0\infty}^{wc}(\alpha) \leq \min\left(1, \frac{2\log_2(2e\alpha)}{\alpha}\right), \tag{6}$$

Note that this bound is independent of probability distributions $\mu_N$.

### 3.2  Piecewise constant functions

Let $PC(d)$ denote the class of piecewise constant binary functions on the unit segment $[0,1)$ with up to $d \geq 0$ discontinuities and with their values defined as 1 at all these discontinuity points. We consider here the learning sequence $\mathcal{L}_N = ([0,1), \mu_N, PC(d_N), t_N)$ where $\mu_N$ is any continuous probability distributions on $[0,1)$, $d_N$ is a monotonic sequence of positive integers diverging to $\infty$ and targets $t_N \in PC(d_{t_N})$ are such that the limit $\delta_t \overset{def}{=} \lim_{N \to \infty} \frac{d_N}{d_{t_N}}$ exists. (Without loss of generality we can assume that all $\mu_N$ are the uniform distribution on $[0,1)$.)

For this learning sequence the following can be established.

**Claim 1.** *The following function defined for $\alpha > 1$ and $0 \leq x \leq 1$ as*

$$\varphi(\alpha, x) \overset{def}{=} -\alpha(1-x)\mathcal{H}\left(\frac{1+\delta_t}{2\alpha(1-x)}\right) - \alpha x \mathcal{H}\left(\frac{1+\delta_t}{2\alpha x}\right) + \alpha \mathcal{H}(x) \quad \text{for } 2\alpha x(1-x) > 1 \,,$$

*and as 0, otherwise, satisfies assumption (3) with respect to the scaling $\tau_N \overset{def}{=} d_N$.*

**Claim 2.** *The following two sided bound on the learning curve holds:*

$$\frac{1}{2\alpha^-}\left(1 + \ln(2\alpha^-)\right) \leq \epsilon_{\lambda\infty}^{wc}(\alpha) \leq \frac{1}{2\alpha^+}\left(1 + \ln(2\alpha^+)\right) \tag{7}$$

*for $\alpha > 1$, $0 \leq \lambda \leq 1$ and $0 \leq \delta_t \leq \alpha\lambda/2$, where $\alpha^- \overset{def}{=} \frac{\alpha}{1-\delta_t+\alpha\lambda/2}$, $\alpha^+ \overset{def}{=} \frac{\alpha}{1+\delta_t+\alpha\lambda}$.*

We outline the main steps of proof of these two claims now.

For Claim 1 we start with a combinatorial argument establishing that in the particular case of constant target

$$\Delta_{i,N}^m = \left\{ \begin{array}{ll} \binom{m-1}{i-1}^{-1} \sum_{j=0}^{d_N/2} \binom{m-i-1}{j-1}\binom{i-1}{j} & \text{for } d + d_t < \min(2i, 2(m-i)), \\ 1 & \text{otherwise.} \end{array} \right.$$

Next we observe that that the above sum equals

$$e^{o(d_N)} \times \max_{0 \le j \le d_N/2} (\binom{m-i}{j}\binom{i}{j}) = e^{\max_{0 \le x \le i/m} \alpha(1-x)\mathcal{H}\left(\frac{1}{2\alpha(1-x)}\right)+\alpha x \mathcal{H}\left(\frac{1}{2\alpha x}\right)+o(d_N)}.$$

This easily gives Claim 1 for constant target ($\delta_t = 0$). Now we observe that this particular case gives an upper bound for the general case (of non-constant target) if we use the "effective" number of discontinuities $d_N + d_{t_N}$ instead of $d_N$.

For Claim 2 we start with the estimate [12, 11]

$$\frac{\lfloor d_N/2\rfloor}{m+1}\left(1 + \sum_{j=\lfloor d_N/2\rfloor+1}^{m+1} \frac{1}{j}\right) \le \epsilon_0^{wc}(m) \le \frac{\lfloor d_N/2\rfloor + 1}{m+1}\left(1 + \sum_{j=\lfloor d_N/2\rfloor+2}^{m+1} \frac{1}{j}\right).$$

derived from the Mauldon result [14] for the constant target $t_N = const$, $m \ge d_N$. This implies immediately the expression

$$\epsilon_{0\infty}^{wc}(\alpha) = \frac{1}{2\alpha}\left(1 + \ln(2\alpha)\right). \tag{8}$$

for the constant target, which extends to the estimate (7) with a straightforward lower and upper bound on the "effective" number of discontinuities in the case of a non-constant target.

### 3.3   Link to multilayer perceptron

Let $MLP^n(w_1)$ denote the class of function from $\mathbf{R}^n$ to $\{0,1\}$ which can be implemented by a multilayer perceptron (feedforward neural network) with $\ge 1$ number of hidden layers, with $w_1$ connections to the first hidden layer and the first hidden layer composed entirely of fully connected, linear threshold logic units (i.e. units able to implement any mapping of the form $(x_1,..,x_n) \mapsto \theta(a_0 + \sum_{i=1}^n a_i x_i)$ for $a_i \in \mathbf{R}$). It can be shown from the properties of Vandermonde determinant (c.f. [7, 8]) that if $f : [0,1) \to \mathbf{R}^n$ is a mapping with coordinates composed of linearly independent polynomials (generic situation) of degree $\le n$, then

$$PC(w_1) = f^* MLP^n(w_1) \stackrel{def}{=} \{h \circ f ; h \in MLP^n(w_1)\}. \tag{9}$$

This implies immediately that all results for learning the class of PC functions in Section 5.2 are applicable (with obvious modifications) to this class of multilayer perceptrons with probability distribution concentrated on the 1-dimensional curves of the form $f([0,1))$ with $f$ as above.

However, we can go a step further. We can extend such a distribution to a continuous distribution on $\mathbf{R}^n$ with support "sufficiently close" to the curve $f([0,1))$,

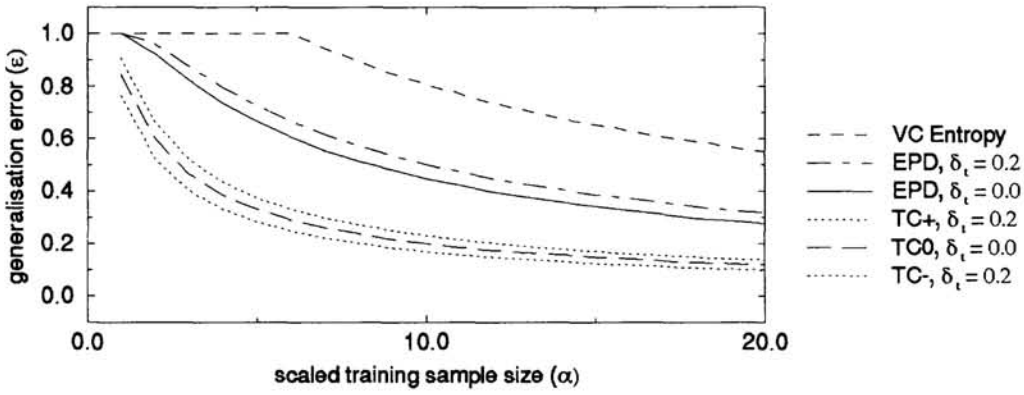

Figure 1: Plots of different estimates for thermodynamic limit of learning curves for the sequence of multilayer perceptrons as in Claim 3 for consistent learning ($\lambda = 0$). Estimates on true learning curve from (7) are for $\delta_t = 0$ ('TC0') and $\delta_t = 0.2$ ('TC+' and 'TC−' for the upper and lower bound, respectively). Two upper bounds of the form (4) from the modified VC-formalism for $\varphi$ as in Claim 1 and $\beta = 1$ are plotted for $\delta_t = 0.0$ and $\delta_t = 0.2$ (marked EPD). For comparison, we plot also the bound (10) based on the VC-entropy; VC bound (5) being trivial for this scaling, $\equiv 1$, c.f. Corollary 2, is not shown.

with changes to the error pattern densities $\Delta_{i,N}^m$, the learning curves, etc., as small as desired. This observation implies the following result:

**Claim 3**  *For any sequence of multilayer perceptrons, $MLP^{n_N}(w_{1N})$, $w_{1N} \to \infty$, there exists a sequence of continuous probability distributions $\mu_N$ on $\mathbf{R}^{n_N}$ with properties as follows. For any sequence of targets $t_N \in MLP^{n_N}(w_{1t_N})$, both Claim 1 and Claim 2 of Section 3.2 hold for the learning sequence $\left(\mathbf{R}^{n_N}, \mu_N, MLP^{n_N}(w_{1N}), t_N\right)$ with scaling $\tau_N \stackrel{def}{=} n_{1N}$ and $\delta_t = \lim_{N \to \infty} w_{1t_N}/w_{1N}$. In particular bound (4) on the learning curve holds for $\varphi$ as in Claim 1.*

**Corollary 2**  *If additionally the number of units in first hidden layer $\mathcal{H}_{1N} \to \infty$, then the thermodynamic limit of VC-bound (5) with respect to the scaling $\tau_N = w_{1N}$ is trivial, i.e. $= 1$ for all $\alpha > 0$.*

**Proof.**  The bound (5) is trivial for $m \leq 12d_N$, where $d_N \stackrel{def}{=} d_{VC}(MLP^{n_N}(w_{1t_N}))$. As $d_N = \Omega(w_{1N} \log_2(\mathcal{H}_{1N}))$ [13, 15] for any *continuous* probability on the input space, this bound is trivial for any $\alpha = \frac{m}{w_{1N}} \leq 12\frac{d_N}{w_{1N}} \to \infty$ if $N \to \infty$. $\square$

There is a possibility that VC dimension based bounds are applicable but fail to capture the true behavior because of their independence from the distribution. One option to remedy the situation is to try a distribution-specific estimate such as VC entropy (i.e. the expectation of the logarithm of the counting function $\Pi_N(x_1, ..., x_m)$ which is the number of dichotomies realised by the perceptron for the $m$-tuple of input points [18]). However, in our case, $\Pi_N(x_1, ..., x_m)$ has the lower bound $2\sum_{i=0}^{\min(w_{1N}/2, m-1)} \binom{m}{i}$, for $x_1, ..., x_m$ in general position, which is virtually the expression from Sauer's lemma with VC-dimension replaced by $w_{1N}/2$. Thus using

VC entropy instead of VC dimension (and Sauer's Lemma) we cannot hope for a better result than bounds of the form (5) with $w_{1N}/2$ replacing VC-dimension resulting in the bound

$$\epsilon_0^{wc}(\alpha) \leq \min(1, \alpha^{-1}\log_2(4e\alpha)) \qquad (\alpha > 1/e) \tag{10}$$

in the thermodynamic limit with respect to the scaling $\tau_N = w_{1N}$. (Note that more "optimistic" VC entropy based bounds can be obtained if prior distribution on hypothesis space is given and taken into account [3].)

The plots of learning curves are shown in Figure 1.

**Acknowledgement.** The permission of Director of Telstra Research Laboratories to publish this paper is gratefully acknowledged.

## Footnotes

[1]Known also as *the partition function* in computational learning theory.

[2] In this paper $\max(S)$, where $S \subset \mathbf{R}$, denotes the maximal element in the closure of $S$, or $\infty$ if no such element exists. Similarly, we understand $\min(S)$.

[3] Note that our learning curve is determined by the worst generalisation error of acceptable hypotheses and in this respect differs from "average generalisation error" learning curves considered elsewhere, e.g. [3, 5].

[4]We recall that $\lfloor x \rfloor$ denotes the largest integer $\leq x$ and $\limsup_{N\to\infty} x_N$ is defined as $\lim_{N\to\infty}$ of the monotonic sequence $N \mapsto \max\{x_1, x_2, ..., x_N\}$. Note that in contrast to the ordinary limit, $\limsup$ always exists.

[5]Note the difference to the concept of error shells used in [4] which are partitions of the finite hypothesis space $H$ according to the generalisation error values. Both formalisms are related though, and the central result in [4], Theorem 4, can be derived from our Theorem 1 below.

# References

[1] A. Blumer, A. Ehrenfeucht, D. Haussler, and M.K. Warmuth. Learnability and the Vapnik-Chervonenkis dimensions. *Journal of the ACM*, **36**:929–965, (Oct. 1989).

[2] T.M. Cover. Geometrical and statistical properties of linear inequalities with applications to pattern recognition. *IEEE Trans. Elec. Comp.*, **EC-14**:326-334, 1965.

[3] D. Hausler, M. Kearns, and R. Shapire. Bounds on the Sample Complexity of Bayesian Learning Using Information Theory and VC Dimension. *Machine Learning*, **14**:83–113, (1994).

[4] D. Haussler, M. Kearns, H.S. Seung, and N. Tishby. Rigorous learning curve bounds from statistical mechanics. In *Proc. COLT'94*, pages 76–87, 1994.

[5] S.B. Holden and M. Niranjan. On the Practical Applicability of VC Dimension Bounds. *Neural Computation*, **7**:1265–1288, 1995).

[6] P. Koiran and E.D. Sontag. Neural networks with quadratic VC-dimension. In *Proc. NIPS 8*, pages 197–203, The MIT Press, Cambridge, Ma., 1996. .

[7] A. Kowalczyk. Counting function theorem for multi-layer networks. In *Proc. NIPS 6*, pages 375–382. Morgan Kaufman Publishers, Inc., 1994.

[8] A. Kowalczyk. Estimates of storage capacity of multi-layer perceptron with threshold logic hidden units. Neural networks, to appear.

[9] A. Kowalczyk and H. Ferra. Generalisation in feedforward networks. Proc. NIPS 6, pages 215–222, The MIT Press, Cambridge, Ma., 1994.

[10] A. Kowalczyk. An asymptotic version of EPD-bounds on generalisation in learning systems. 1996. Preprint.

[11] A. Kowalczyk, J. Szymanski, and R.C. Williamson. Learning curves from a modified VC-formalism: a case study. In *Proc. of ICNN'95* , 2939–2943, IEEE, 1995.

[12] A. Kowalczyk, J. Szymański, P.L. Bartlett, and R.C. Williamson. Examples of learning curves from a modified VC-formalism. Proc. NIPS 8, pages 344–350, The MIT Press, 1996.

[13] W. Maas. Neural Nets with superlinear VC-dimesnion. *Neural Computation*, **6**:877–884, 1994.

[14] J.G. Mauldon. Random division of an interval. *Proc. Cambridge Phil. Soc.*, **47**:331–336, 1951.

[15] A. Sakurai. Tighter bounds of the VC-dimension of three-layer networks. In *Proc. of the 1993 World Congress on Neural Networks*, 1993.

[16] E. Sontag. Shattering all sets of $k$ points in "general position" requires $(k-1)/2$ parameters. Report 96-01, Rutgers Center for Systems and Control, 1996.

[17] V. Vapnik. *Estimation of Dependences Based on Empirical Data*. Springer-Verlag, 1982.

[18] V. Vapnik. *The Nature of Statistical Learning Theory*. Springer-Verlag, 1995.
